# Artificial Olfactory Brain for Mixture Identification

**Mehmet K. Muezzinoglu**[1]    **Alexander Vergara**[1]    **Ramon Huerta**[1]    **Thomas Nowotny**[2]

**Nikolai F. Rulkov**[1]    **Heny D. I. Abarbanel**[1]    **Allen Selverston**[1]    **Mikhail I. Rabinovich**[1]

[1] Institute for Nonlinear Science
University of California San Diego
9500 Gilman Dr., La Jolla, CA, 92093-0402

[2] Centre for Computational Neuroscience and Robotics
Department of Informatics, University of Sussex
Falmer, Brighton, BN1 9QJ, UK

## Abstract

The odor transduction process has a large time constant and is susceptible to various types of noise. Therefore, the olfactory code at the sensor/receptor level is in general a slow and highly variable indicator of the input odor in both natural and artificial situations. Insects overcome this problem by using a neuronal device in their Antennal Lobe (AL), which transforms the identity code of olfactory receptors to a spatio-temporal code. This transformation improves the decision of the Mushroom Bodies (MBs), the subsequent classifier, in both speed and accuracy. Here we propose a rate model based on two intrinsic mechanisms in the insect AL, namely integration and inhibition. Then we present a MB classifier model that resembles the sparse and random structure of insect MB. A local Hebbian learning procedure governs the plasticity in the model. These formulations not only help to understand the signal conditioning and classification methods of insect olfactory systems, but also can be leveraged in synthetic problems. Among them, we consider here the discrimination of odor mixtures from pure odors. We show on a set of records from metal-oxide gas sensors that the cascade of these two new models facilitates fast and accurate discrimination of even highly imbalanced mixtures from pure odors.

## 1   Introduction

Odor sensors are diverse in terms of their sensitivity to odor identity and concentrations. When arranged in parallel arrays, they may provide a rich representation of the odor space. Biological olfactory systems owe the bulk of their success to employing a large number of olfactory receptor neurons (ORNs) of various phenotypes. However, chemo-diversity comes at the expense of two pressing factors, namely response time and reproducibility, while fast and accurate processing of chemo-sensory information is vital for survival not only in natural, but also in many artificial situations, including security applications.

Identifying and quantifying an odor accurately in a short time is an impressive characteristic of insect olfaction. Given that there are approximately tens of thousands of ORNs sending slow and noisy messages in parallel to downstream olfactory layers, in order to account for the observed recognition performance, a computationally non-trivial process must be taking place along the insect olfactory pathway following the transduction. The two stations responsible for this processing are the Antennal Lobe (AL) and the Mushroom Bodies (MBs). The former acts as a signal conditioning / feature extraction device and the latter as an algebraic classifier.

Our motivation in this study is the potential for skillful feature extraction and classification methods by insect olfactory systems in synthetic applications, which also deal with slow and noisy sensory data. The particular problem we address is the discrimination of two-component odor mixtures from

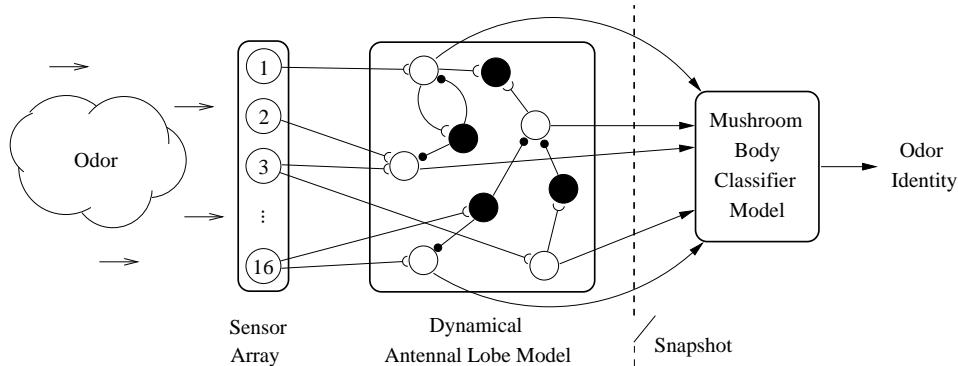

Figure 1: The considered biomimetic framework to identify whether an applied gas is a pure odor or a mixture. The input is transduced by 16 parallel metal-oxide gas sensors of different type generating slow and noisy resistance time series. The signal conditioning in the antennal lobe is achieved by the interaction of an excitatory Projection Neuron (PN) population (white nodes) with an inhibitory Local Neurons (LNs, black nodes). The outcomes of AL processing is read from the PNs and classified in the Mushroom Body, which is trained by a local Hebbian rule.

pure odors in a three-class classification setting. The problem is nontrivial when concentrations of mixture components are imbalanced. It becomes particularly challenging when the overall mixture concentration is small. We treat the problem on two mixture datasets recorded from metal-oxide gas sensors (included in the supplementary material).

We propose in the next section a dynamical rate model mimicking the AL's signal conditioning function. By testing the model first with a generic Support Vector Machine (SVM) classifier, we validate the substantial improvement that AL adds on the classificatory value of raw sensory signal (Section 2). Then, we introduce a MB-like classifier to substitute for the SVM and complete the biomimetic framework, as outlined in Fig. 1. The model MB exploits the structural organization of the insect MB. Its plasticity is adjusted by a local Hebbian learning procedure, which is gated by a binary learning signal (Section 3). Some concluding remarks are given in Section 4.

## 2   The Antennal Lobe

### 2.1   Insect Antennal Lobe Outline

The Antennal Lobe is a spatio-temporal encoder for ORN signals that include time in coding space. Some of its qualitative properties are apparent from the input-output perspective, without requiring much insight into its physiology. A direct analysis of spiking rates in raw ORN responses and in the AL output [1] shows that in fruit fly AL maps ORN output to a low dimensional feature space while providing lower variability in responses to the same odor type (reducing within-class scatter) and longer average distance between responses for different odors (boosting between-class scatter). These observations constitute sufficient evidence that a realistic AL model should be sought within the class of nonlinear filters.

Another remarkable achievement of the AL shows itself in terms of recognition time. When subjected to a constant odor concentration, the settling time of ORN activity is on the order of hundreds of milliseconds to seconds [3], whereas recognition is known to occur earlier [7]. This is a clear indicator that the AL makes extensive use of the ORN transient, since instantaneous activity is less odor-specific in transient than it is in during the steady state. To provide high accuracy under such a temporal constraint, the classificatory information during this period must be somehow accumulated, which means that AL has to be a dynamical system, utilizing memory.

It is the cooperation of these filtering and memory mechanisms in the AL that expedites and consolidates the decision made in the subsequent classifier.

Strong experimental evidence suggests that the insect AL representation of odors is a transient, yet reproducible, spatio-temporal encoding [8]. The AL is a dynamical network that is formed by the coupling of an excitatory neuron population (projection neurons, PNs) with an inhibitory one

(local neurons, LNs). It receives input from glomeruli, junctions of synapses that group the ORNs according to the receptor gene they express. The fruit fly has about 50 glomeruli as chemotopic clusters of synapses from nearly $50,000$ ORNs. There is no consensus on the functional role of this convergence beyond serving as an input terminal to AL, which is certainly an active processing layer. In the analogy we are building here (c.f. Fig. 1), the 16 artificial gas sensors actually correspond to glomeruli (rather than individual ORNs) so that the AL has direct access to sensor resistances.

We suggest that the two key principles underlying the AL's information processing are decorrelation (filtering) and integration (memory), which can be unified on a dynamical system. The filter property provides selectivity, while the integrator accumulates the refined information on trajectories. This setting is capable of evaluating the transient portion of the sensory signal effectively.

An instantaneous value read from a receptor early in the transduction process is considered as immature, failing to convey a consistently high classificatory value by its own. Nevertheless, the ORN transient as an interval indeed offers unique features to expedite the classification. In particular, the novelty gained due to observing consecutive samples during the transient is on average greater than the informational gain obtained during the steady-state. Hence, newly observed samples of the receptor transients are likely to contribute to the cumulative classificatory information base formed so far, whereas the informational entropy vanishes as the signal reaches the steady-state. As a device that extracts and integrates odor-specific information in ORN signals, the AL provides an enriched transient to the subsequent MB so that it can achieve accurate classification early in the odor period.

We also note that there have been efforts, e.g., [9, 10] to illustrate the sharpening effect of inhibition in the olfactory system. However, to the best of our knowledge, the approach we present here is the first to formulate the temporal gain due to AL processing.

## 2.2 The Model

The model AL is comprised of a population of PNs that project from the AL to downstream processing. The neural activity corresponding to the rate of action potential generation of the biological neurons is given by $x_i(t)$, $i = 1, 2, ..., N_E$, for the $N_E$ neurons in the PN population. There are also $N_I$ interneurons or LNs whose activity is $y_i(t)$; $i = 1, 2, ..., N_I$.

The rate of change in these activities is stimulated by a weighted sum over both populations and a set of input signals $S_i^E(t)$ and $S_i^I(t)$ indicating the activity in the glomeruli stimulating the PNs and the LNs, respectively. In addition, each population receives noise from the AL environment. Our formulation of these ideas is through a Wilson-Cowan-like population model [11]

$$\beta_i^E \frac{dx_i(t)}{dt} = K_i^E \cdot \Theta \left( -\sum_{j=1}^{N_I} w_{ij}^{EI} y_j(t) + g_{inp}^E S_i^E(t) \right) - x_i(t) + \mu_i^E(t), \ \ i \in 1, \ldots, N_E,$$

$$\beta_i^I \frac{dy_i(t)}{dt} = K_i^I \cdot \Theta \left( \sum_{j=1}^{N_E} w_{ij}^{IE} x_j(t) + g_{inp}^I S_i^I(t) \right) - y_i(t) + \mu_i^I(t), \ \ i \in 1, \ldots, N_I.$$

The superscripts $E$ and $I$ stand for excitatory and inhibitory populations. The matrix elements $w_{ij}^{XY}$, $X, Y \in \{E, I\}$ are time-independent weights quantifying the effect from units of type $Y$ to units of type $X$. $S_i^X(t)$ is the external input to $i$-th unit from a glomerulus (odor sensor) weighted by coupling strength $g_{inp}^X$. $\mu_i^Y$ is an additive noise process and $\Theta(\cdot)$, the unit-ramp activation function: $\Theta(u) = 0$ for $u < 0$, and $\Theta(u) = u$, otherwise. The gains $K_i^E, K_i^I$ and time constants $\beta_i^E, \beta_i^I$ are fixed for an individual unit but vary across PN and LN populations.

The network topology is formed through a random process of Bernoulli type:

$$w_{ij}^{XY} = g_Y \cdot \begin{cases} 1 & , \ \text{with probability } p^{XY} \\ 0 & , \ \text{with probability } 1 - p^{XY} \end{cases}$$

where $g_Y$ is a fixed coupling strength. $p^{XY}$ is a design parameter to be chosen by us.

Each unit, regardless of its type, accepts external input from exactly one sensor in the form of raw resistance time series. This sensor is assigned randomly among all 16 available sensors, ensuring that all sensors are covered[1].

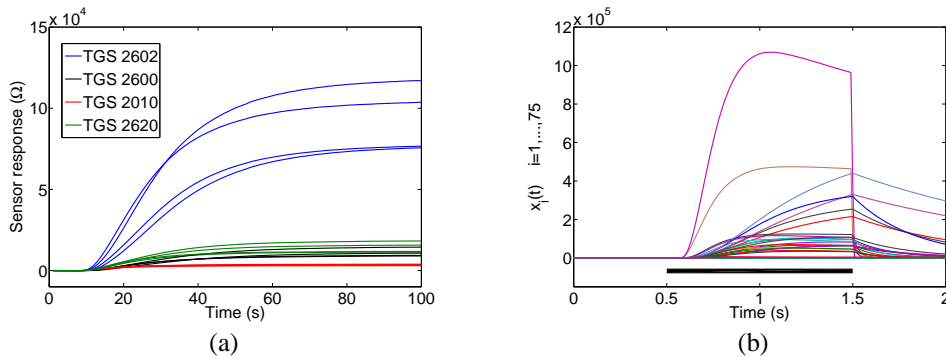

(a)  (b)

Figure 2: (a) A record from Dataset 1, where 100ppm acetaldehyde was applied to the sensor array for $0 \leq t \leq 100$s. Offsets are removed from the time-series. Curve labels indicate the sensor types. (b) Activity of $N_E = 75$ excitatory PN units of the sample AL model in response to the (time-scaled version of) record shown on panel (a). The conductances are selected as $(g_E, g_I) = (10^{-6}, 9 \cdot 10^{-6})$ and other parameters as given in text. Bar indicates the odor period.

For the mixture identification problem of this study, we consider a network with $N_E = N_I = 75$ and $g_{\text{inp}}^E = g_{\text{inp}}^I = 10^{-2}$. The probabilities used in the generative Bernoulli process are fixed at $p_{IE} = p_{EI} = 0.5$. The synaptic conductances $g_E$ and $g_I$ are optimized for the particular classification instance through the brute force search described below. The gains $K_i^E, K_j^I$ and the time-scales $\beta_i^E, \beta_j^I, i = 1, \ldots, N_E, j = 1, \ldots, N_I$ are drawn independently from exponential distributions with $\lambda_K = 7.5$ and $\lambda_\beta = 0.5$, respectively. Following construction, the initial condition of each unit is taken as zero and $\mu$ is taken as a white noise process with variance $10^{-4}$ independently for each unit. We perform the simulation of the 150-dimensional Wilson-Cowan dynamics by 5/6 Runge-Kutta integration with variable step size where the error tolerance is set to $10^{-15}$.

Although the considered network structure can accommodate limit cycles and strange attractors, the selected range of parameters yield a fixed point behavior. We confirm this in all simulations with the selected parameter values, both during and after the sensory input (odor) period (see Fig. 2(b)).

## 2.3 Validation

We consider the activity in PN population as the only piece of information regarding the input odor that is passed on to higher-order layers of the olfactory system. Access to this activity by those layers can be modeled as instantaneous sampling of a selected brief window of temporal behavior of PNs [7]. Therefore, the recognition system in our model utilizes such snapshots from the spiking activity in the excitatory population $x_i(t)$. A snapshot is passed as the feature vector to the classifier; it is comprised of an $N_E$-dimensional fixed vector taken as a sample from the states $x_1, \ldots, x_{N_E}$ at a particular time $t_s$.

### 2.3.1 Dataset

The model is driven by responses recorded from 16 metal-oxide gas sensors in parallel. We have made 80 recordings and grouped them into two sets based on vapor concentration: records for 100ppm vapor in Dataset 1 and 50ppm in Dataset 2. Each dataset contains 40 records from three classes: 10 pure acetaldehyde, 10 pure toluene, and 20 mixture records. The mixture class contains records from imbalanced acetaldehyde-toluene mixtures with 96%-4%, 98%-2%, 2%-98%, and 4%-96% partial concentrations, five from each. Hence, we have two instances of the mixture identification problem in the form of three-class classification. See the supplementary material for details on measurement process.

We removed the offset from each sensor record and scaled the odor period to 1s. This was done by mapping the odor period, which has fixed length of 100s in the original records, to 1s by re-indexing the time series. These one-second long raw time series, included in the supplementary material, constitute the pool of raw inputs to be applied to the AL network during the time interval $0.5 \leq t \leq 1.5$s. The input is set to zero outside of this odor period. See Fig. 2 for a sample record and the AL network's response to it. Note that, although we apply the network to pre-recorded data in simulations, the general scheme is causal, thus can be applied in real-time.

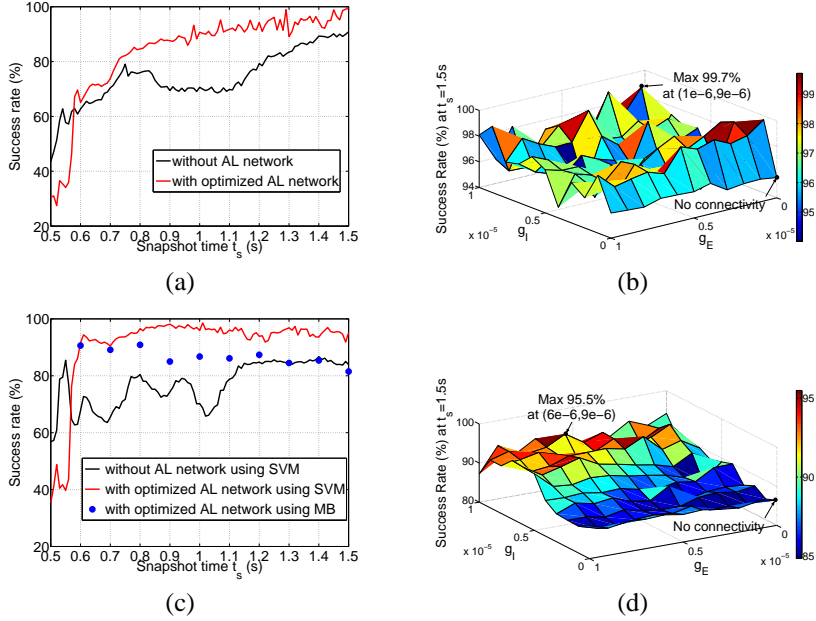

Figure 3: (a) Estimated correct classification profile versus snapshot time $t_s$ during the normalized odor period for Dataset 1. The red curve is the classification profile due to the proposed AL network, which has the fixed sample topology with $(g_E, g_I) = (10^{-6}, 9 \cdot 10^{-6})$. The black baseline profile is obtained by discarding AL and directly classifying snapshots from raw sensor responses by SVM. (b) Correct classification rates extracted by a sweep through $g_E, g_I$ using Dataset 1. Panels (c) & (d) show the results for Dataset 2, where the best pair is determined as $(g_E, g_I) = (6 \cdot 10^{-6}, 9 \cdot 10^{-6})$.

### 2.3.2 Adjustment of AL Network and Performance Evaluation

To reveal the signal conditioning performance of the stand-alone AL model, we first interface it with an established classifier. We use a Support Vector Machine (SVM) classifier with linear kernel to map the snapshots from PN activity to odor identity. This choice is due to the parameter-free design that rules out the possibility of over-fitting. The classifier is realized by the publicly available software LibSVM [2].

Due to the wide diversity of PNs and LNs in terms of their time scales $\beta$ and gains $K$, the performance of the network is highly sensitive to the agreement between the outcome of the generative process and the choice of parameters $g_E$ and $g_I$. Therefore, it is not possible to give a one-size-fits-all value for these. Instead, we have generated one sample network topology via the Bernoulli process described above and customized $g_E$ and $g_I$ for it on each problem. For reproducibility, this topology is provided in the supplementary material. Comparable results can be obtained with other topologies but possibly with different $g_E, g_I$ values than the ones reported below.

The validation is carried out in the following way: First we set the classification problem (i.e., select Dataset 1 or 2) and fix $g_E = g_I = 0$ (suppress the connectivity). We present each record in the dataset to the network and then log the network response from excitatory population in the form of $N_E$ simultaneous time series (see Fig. 2). Then, at each percentile of the odor period $t_s \in \{0.5 + k/100\}_{k=0}^{100}$, we take a snapshot from each $N_E$-dimensional time series and label it by the odor identity (pure acetaldehyde, pure toluene, or mixture). We use randomly selected $80\%$ of the resulting data in training the SVM classifier and keep the remaining $20\%$ for testing it. We record the rate of correct classification on the test data. The train-test stage is repeated $1000$ times with different random splits of labelled data. The average correct classification rate is assigned as the performance of the AL model at that $t_s$. The classification profile versus time is extracted when the $t_s$ sweep through the odor period is complete.

To maximize the performance over conductances $g_E$ and $g_I$, we further perform a sweep through a range of these parameters by repeating the above procedure for each combination of $g_E, g_I$. Fig-

ure 3 (a) shows the classification profile for the best pair encountered along the parameter sweep $g_E, g_I \in \{k/100\}_{k=0}^{100}$. This pair is determined as the one maximizing classification success rate when samples from the end of odor period is used $t_s = 1.5$. Note that these optimum values are problem-specific. For the two instances considered in this work, we mark them by the peaks of the surfaces in Fig. 3 (b) and (d).

Dataset 1 induces an easier instance of the identification problem toward the end of odor period, which can be resolved reasonably well using raw sensor data at the steady state. Therefore, the gain over baseline due to AL processing is not so significant in later portions of the odor period for Dataset 1. Also observe from panels (b) and (d) that, when dealing with Dataset 1, the conductance values are less decisive than they are for Dataset 2. Again, this is because the former is an easier problem when the sensors reach the steady-state at $t_s = 1.5$s, where almost all conductance within the swept range ensures $> 95\%$ performance. The relative difficulty of the problem in Dataset 2 manifests itself as the fluctuations in the baseline performance. We see in Fig. 3(c) that there are actually periods early in the period where the raw sensor data can be fairly indicative of the class information; however, it is not possible to predict these intervals in advance. It should also be noted that some of these peaks in baseline performance, at least the very first one near $t_s = 0.55$s, are artifacts (due to classification of pure noise) since we know that there is hardly any vapor in the measurement chamber during that period (see Fig. 2(a) and other records in supplementary material). In any case, in both problems, the suggested AL dynamics (with adjusted parameters) contributes substantially to the classification performance during the transient of the sensory signal. This makes early decisions of the classifier substantially and consistently more accurate with respect to the baseline classification.

Having established the contribution of the AL network to classification, our goal in the remainder of the paper is to replace the unbiased SVM classifier by a biologically plausible MB model, while preserving the performance gain seen in Fig. 3.

## 3 Mushroom Body Classifier

The MBs of insects employ a large number of identical small intrinsic cells, the so-called Kenyon cells, and fewer output neurons in the MB lobes. It has been observed that, unlike in the AL, the activity in the KCs is very sparse, both across the population and for individual cells over time. Theoretical work suggests that a large number of cells with sparse activity enables efficient classification with random connectivity [4]. The power of this architecture lies in its versatility: The connectivity is not optimized for any specific task and can, therefore, accommodate a variety of input types.

### 3.1 The Model

The insect MB consists of four crucial elements (see Fig. 4): i) a nonlinear expansion from the AL representation at the final stage, $\mathbf{x}$, that resembles the connectivity from the Antennal Lobe to the MBs, ii) a gain control in the MB to achieve a uniform level of sparse activity the KCs, $\mathbf{y}$, iii) a classification phase, where the connections from the KCs to the output neurons, $\mathbf{z}$, are modified according to a Hebbian learning rule, and iv) a learning signal that determines when and which output neuron's synapses are reinforced.

It has been shown in locusts that the activity patterns in the AL are practically discretized by a periodic feedforward inhibition onto the MB calyces and that the activity levels in KCs are very low [7]. Based on the observed discrete and sparse activity pattern in insect MB, we choose to represent KC units as simple algebraic McCulloch-Pitts 'neurons.' The neural activity values taken by this neural model are binary (0 = no spike and 1 = spike): $\mu_j = \Phi\left(\sum_{i=1}^{N_E} c_{ji} x_i - \theta^{KC}\right)$ $j = 1, 2, ..., N_{KC}$. The vector $\mathbf{x}$ is the representation of the odor that is received as a snapshot from the excitatory PN units of AL model. The components of the vector $\mathbf{x} = (x_1, x_2, ..., x_{N_E})$ are the direct values obtained by integration of the ODE of the AL model described above. The KC layer vector $\mu$ is $N_{KC}$ dimensional. $c_{ij} \in \{0, 1\}$ are the components of the connectivity matrix which is $N_E \times N_{KC}$ in size. The firing threshold $\theta^{KC}$ is integer number and $\Phi(\cdot)$ is the Heaviside function.

The connectivity matrix $[c_{ji}]$ is determined randomly by an independent Bernoulli process. Since the degree of connectivity from the input neurons to the KC neurons did not appear to be critical for the performance of the system, we made it uniform by setting the connection probability as $p_c = 0.1$. It, nevertheless, seems advisable to ensure in the construction that the input-to-KC layer mapping is bijective to avoid loss of information. We performed this check during network construction. All other parameters of the KC layer are then assigned admissible values uniformly randomly and fixed.

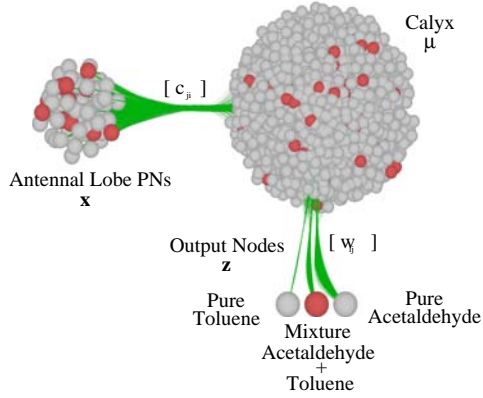

Figure 4: Suggested MB model for classifying the AL output. The first layer of connections from AL to calyx are set randomly and fixed. The plasticity of the output layer is due to a binary learning signal that rewards the weights of output units responding to the correct stimulus.

Although the basic system described so far implements the divergent (and static) input layer observed in insect calyx, it is very unstable against fluctuations in the total number of active input neurons due to the divergence of connectivity. This is an obstacle for inducing sparse activity at KC level. One mechanism suggested to remove this instability is gain control by feedforward inhibition. For our purposes, we impose a number $n_{\text{KC}}$ of simultaneously active KCs, and admit the firing of only the top $n_{\text{KC}} = N_{\text{KC}}/5$ neurons that receive the most excitation in the first layer.

The fan-in stage of projections from the KCs to the extrinsic MB cells in the MB lobes is the hypothesized locus of learning. In our model, the output units in the MB lobes are again McCulloch-Pitts neurons: $z_l = \Phi\left(\sum_{j=1}^{N_{\text{KC}}} w_{lj} \cdot \mu_j - \theta^{\text{LB}}\right)$, $l = 1, 2, ..., N_{\text{LB}}$. Here, the index LB denotes the MB lobes. The output vector $\mathbf{z}$ of the MB lobes has dimension $N_{\text{LB}}$ (equals 3 in our problem) and $\theta^{\text{LB}}$ is the threshold for the decision neurons in the MB lobes. The $N_{\text{LB}} \times N_{\text{KC}}$ connectivity matrix $w_{lj}$ has integer entries. Similar to the above-mentioned gain control, we allow only the decision neuron that receives the highest synaptic input to fire. These synaptic strengths $w_{lj}$ are subject to changes during learning according to a Hebbian type plasticity rule described next.

### 3.2 Training

The hypothesis of locating reinforcement learning in mushroom bodies goes back to Montague and collaborators [6]. Every odor class is associated with an output neuron of the MB, so there are three output nodes firing for either pure toluene, pure acetaldehyde, or mixture type of input. The plasticity rule is applied on the connectivity matrix $W$, whose entries are randomly and independently initialized within $[0, 10]$. The exact initial distribution of weights have no significant impact on the resulting performance nor on the learning speed.

During learning, the inputs are presented to the system in an arbitrary order. The entries of the connectivity matrix at the time of the $n$th input are denoted by $w_{lj}(n)$. When the next training input with label $\ell$ is applied, then the weight $w_{\ell j}$ is updated by the rule $w_{\ell j}(n+1) = H\left(z_\ell, \mu_j, w_{\ell j}(n)\right)$, where $H(z, \mu, w) = w + 1$ when $z = 1$, $\mu = 1$; and 0, otherwise. This learning rule strenghtens a synaptic connection with probability $p_+$ if presynaptic activity is accompanied by postsynaptic activity. To facilitate learning during the training phase, the 'correct' output neuron $\ell$ is forced to fire for an input with label $\ell$, while the rest are kept silent. This is provided by pulling down the threshold $\theta_\ell^{\text{LB}}$, unless neuron $\ell$ is already firing for such input. Learning is terminated when the performance (correct classification rate) converges.

### 3.3 Validation

Using Dataset 2, we applied the proposed MB model with $N_{\text{KC}} = 10,000$ KCs at the output of the sample AL topology having the same parameters reported in Section 2. For $p_+ = p_- = 1$, we trained the output layer of MB using the labelled AL outputs sampled at 10 points in the odor period. The mean correct classification rate over 20 splits of the labelled snapshots (five-fold cross-validation) are shown in Fig.3(c) as blue dots. With respect to the red curve on the same panel, which was obtained by the (maximum-margin) SVM classifier, a slight reduction in the generalization capability is visible. Nevertheless, the MB classifier in its current form still exploits the superior job of AL over baseline classification during transient, while mimicking two essential features of the biological MB, namely sparsity in KC-layer and incremental local learning in MB lobes. The implementation details and parameters of the MB model are provided in the supplementary material.

## 4 Conclusions

We have presented a complete odor identification scheme based on the key principles of insect olfaction, and demonstrated its validity in discriminating mixtures of odors from pure odors using actual records from metal-oxide gas sensors.

The bulk of the observed performance is due to the AL, which is a dynamical feature extractor for slow and noisy chemo-sensory time series. The cooperation of integration (accumulation) mechanism and sharpening filter enabled by inhibition leave an almost linearly separable problem for the subsequent classifier. The proposed signal conditioning scheme can be considered as a mathematical image of reservoir computing [5]. For this simplified classification task, we have also suggested a bio-inspired MB classifier with local Hebbian plasticity. By exploiting the dynamical nature of the AL stage and the sparsity in MB representation, the overall model provides an explanation for the high speed and accuracy of odor identification in insect olfactory processing.

For future study, we envision an improvement on the MB classification performance, which has been explored here to be slightly worse than linear SVM. We think that this can be done without compromising biological plausibility, by imposing mild constraints on the KC-level generative process.

The mixture identification problem investigated here is in general more difficult than the traditional problem of discriminating pure odors, since the mixture class can be made arbitrarily close to the pure odor classes. The classification performance attained here is promising for other mixture-related problems that are among the hardest in the field of artificial olfaction.

**Acknowledgments**

This work was supported by the MURI grant ONR N00014-07-1-0741.

## Footnotes

[1]It is assumed that $N_E + N_I > 16$.

## References

[1] V. Bhandawat, S. R. Olsen, N. W. Gouwens, M. L. Schlief, and R. I. Wilson. Sensory processing in the drosphila antennal lobe increases reliability and separability of ensemble odor representations. *Nature Neuroscience*, 10:1474–1482, 2007.

[2] C.C. Chang and C. J. Lin. LibSVM - A library for support vector machines, v2.85, 2007.

[3] M de Bruyne, P. J. Clyne, and J. R. Carlson. Odor coding in a model olfactory organ: The Drosophila maxillary palp. *Journal of Neuroscience*, 11:4520–4532, 1999.

[4] R. Huerta, T. Nowotny, M. Garcia-Sanchez, H. D. I. Abarbanel, and M. I. Rabinovich. Learning classification in the olfactory system of insects. *Neural Computation*, 16:1601–1640, 2004.

[5] W. Maass, T. Natschlaeger, and H. Markram. Real-time computing without stable states: A new framework for neural computation based on perturbations. *Neural Computation*, 14:2531–2560, 2002.

[6] P. R. Montague, P. Dayan, C. Person, and T. J. Sejnowski. Bee foraging in uncertain environments using predictive Hebbian learning. *Nature*, 337:725–728, 1995.

[7] J. Perez-Orive, O. Mazor, G. C. Turner, S. Cassenaer, R. I. Wilson, and G. Laurent. Oscillations and sparsening of odor representations in the mushroom body. *Science*, 297:359–365, 2002.

[8] M. I. Rabinovich, R. Huerta, and G. Laurent. Transient dynamics for neural processing. *Science*, 321:48–50, 2008.

[9] B. Raman and R. Gutierrez-Osuna. Chemosensory processing in a spiking model of the olfactory bulb: Chemotopic convergence and center surround inhibition. In L. K. Saul, Y. Weiss, and L. Bottou, editors, *NIPS 17*, pages 1105–1112. MIT Press, Cambridge, MA, 2005.

[10] M. Schmuker and G. Schneider. Processing and classification of chemical data inspired by insect olfaction. *Proc. Nat. Acad. Sci.*, 104:20285–20289, 2007.

[11] H. R. Wilson and J. D. Cowan. A mathematical theory of the functional dynamics of cortical and thalamic nervous tissue. *Kybernetik*, 13:55–80, 1973.

